# Bounding Performance Loss in Approximate MDP Homomorphisms

**Jonathan J. Taylor**
Dept. of Computer Science
University of Toronto
Toronto, Canada, M5S 3G4
jonathan.taylor@utoronto.ca

**Doina Precup**
School of Computer Science
McGill University
Montreal, Canada, H3A 2A7
dprecup@cs.mcgill.ca

**Prakash Panangaden**
School of Computer Science
McGill University
Montreal, Canada, H3A 2A7
prakash@cs.mcgill.ca

## Abstract

We define a metric for measuring behavior similarity between states in a Markov decision process (MDP), which takes action similarity into account. We show that the kernel of our metric corresponds exactly to the classes of states defined by MDP homomorphisms (Ravindran & Barto, 2003). We prove that the difference in the optimal value function of different states can be upper-bounded by the value of this metric, and that the bound is tighter than previous bounds provided by bisimulation metrics (Ferns et al. 2004, 2005). Our results hold both for discrete and for continuous actions. We provide an algorithm for constructing approximate homomorphisms, by using this metric to identify states that can be grouped together, as well as actions that can be matched. Previous research on this topic is based mainly on heuristics.

## 1 Introduction

Markov Decision Processes (MDPs) are a very popular formalism for decision making under uncertainty (Puterman, 1994). A significant problem is computing the optimal strategy when the state and action space are very large and/or continuous. A popular approach is *state abstraction*, in which states are grouped together in partitions, or aggregates, and the optimal policy is computed over these. Li et al. (2006) provide a nice comparative survey of approaches to state abstraction. The work we present in this paper bridges two such methods: bisimulation-based approaches and methods based on MDP homomorphisms.

Bisimulation is a well-known, well-studied notion of behavioral equivalence between systems (Larsen & Skou, 1991; Milner, 1995) which has been specialized for MDPs by Givan et al (2003). In recent work, Ferns et al. (2004, 2005, 2006) introduced (pseudo)metrics for measuring the similarity of states, which provide approximations to bisimulation. One of the disadvantages of bisimulation and the corresponding metrics is that they require that the behavior matches for exactly the same actions. However, in many cases of practical interest, actions with the exact same label may not match, but the environment may contain symmetries and other types of special structure, which may allow correspondences between states by matching their behavior with *different* actions. This idea was formalized by (Ravindran & Barto, 2003) with the concept of MDP homomorphisms. MDP homomorphisms specify a map matching equivalent states as well as equivalent actions in such states. This matching can then be used to transfer policies between different MDPs. However, like any equivalence relations in probabilistic systems, MDP homomorphisms are brittle: a small change in the transition probabilities or the rewards can cause two previously equivalent state-action pairs to become distinct. This implies that such approaches do not work well in situations in which the model of the system is estimated from data. As a solution to this problem, Ravindran & Barto (2004) proposed using *approximate homomorphisms*, which allow aggregating states that are not exactly equivalent. They define an MDP over these partitions and quantify the approximate loss resulting from using this MDP, compared to the original system. As expected, the bound depends on

the quality of the partition. Subsequent work (e.g. Wolfe & Barto, 2006) constructs such partitions heuristically.

In this paper, we attempt to construct provably good, approximate MDP homomorphisms from first principles. First, we relate the notion of MDP homomorphisms to the concept of lax bisimulation, explored recently in the process algebra literature (Arun-Kumar, 2006). This allows us to define a metric on states, similarly to existing bisimulation metrics. Interestingly, this approach works both for discrete and for continuous actions. We show that the difference in the optimal value function of two states is bounded above by this metric. This allows us to provide a state aggregation algorithm with provable approximation guarantees. We illustrate empirically the fact that this approach can provide much better state space compression than the use of existing bisimulation metrics.

## 2 Background

A finite Markov decision process (MDP) is a tuple $\langle S, A, P, R \rangle$, where $S$ is a finite set of states, $A$ is a set of actions, $P : S \times A \times S \to [0,1]$ is the transition model, with $P(s, a, s')$ denoting the probability of transition from state $s$ to $s'$ under action $a$, and $R : S \times A \to \mathbb{R}$ is the reward function with $R(s, a)$ being the reward for performing action $a$ in state $s$. For the purpose of this paper, the state space $S$ is assumed to be finite, but the action set $A$ could be finite or infinite (as will be detailed later). We assume without loss of generality that rewards are bounded in $[0, 1]$.

A deterministic policy $\pi : S \to A$ specifies which action should be taken in every state. By following policy $\pi$ from state $s$, an agent can expect a value of $V^\pi(s) = E(\sum_{t=1}^{\infty} \gamma^{t-1} r_t | s_0 = s, \pi)$ where $\gamma \in (0,1)$ is a discount factor and $r_t$ is the sample reward received at time $t$. In a finite MDP, the optimal value function $V^*$ is unique and satisfies the following formulas, known as the Bellman optimality equations:

$$V^*(s) = \max_{a \in A} \left( R(s, a) + \gamma \sum_{s'} P(s, a, s') V^*(s') \right), \forall s \in S$$

If the action space is continuous, we will assume that it is compact, so the max can be taken and the above results still hold (Puterman, 1994). Given the optimal value function, an optimal policy is easily inferred by simply taking at every state the greedy action with respect to the one-step-lookahead value. It is well known that the optimal value function can be computed by turning the above equation into an update rule, which can be applied iteratively.

Ideally, if the state space is very large, "similar" states should be grouped together in order to speed up this type of computation. Bisimulation for MDPs (Givan et al., 2003) is a notion of behavioral equivalence between states. A relation $E \subseteq S \times S$ is a *bisimulation relation* if:

$$sEu \Leftrightarrow \forall a.(R(s, a) = R(u, a) \text{ and } \forall X \in S/E.Pr(X|s, a) = Pr(X|u, a))$$

where $S/E$ denotes the partition of $S$ into $E$-equivalent subsets of states. The relation $\sim$ is the union of all bisimulation relations and two states in an MDP are said to be *bisimilar* if $s \sim u$. From this definition, it follows that bisimilar states can match each others' actions to achieve the same returns. Hence, bisimilar states have the same optimal value (Givan et al., 2003). However, bisimulation is not robust to small changes in the rewards or the transition probabilities.

One way to avoid this problem is to quantify the similarity between states using a (pseudo)-metric. Ferns et al. (2004) proposed a *bisimulation metric*, defined as the least fixed point of the following operator on the lattice of 1-bounded metrics $d : S \times S \to [0, 1]$:

$$G(d)(s, t) = \max_a (c_r |R(s, a) - R(u, a)| + c_p K(d)(P(s, a, \cdot), P(u, a, \cdot)) \tag{1}$$

The first term above measures reward similarity. The second term is the Kantorovich metric between the probability distributions of the two states. Given probability distributions $P$ and $Q$ over the state space $S$, and a semimetric $d$ on $S$, the Kantorovich metric $K(d)(P, Q)$ is defined by the following linear program:

$$\max_{v_i} \sum_{i=1}^{|S|} (P(s_i) - Q(s_i)) v_i \text{ subject to: } \forall i, j.v_i - v_j \le d(s_i, s_j) \text{ and } \forall i.0 \le v_i \le 1$$

which has the following equivalent dual program:

$$\min_{\lambda_{kj}} \sum_{k,j=1}^{|S|} \lambda_{kj} d(s_k, s_j) \text{ subject to: } \forall k. \sum_j \lambda_{kj} = P(s_k), \forall j. \sum_k \lambda_{kj} = Q(s_j) \text{ and } \forall k, j.\lambda_{kj} \ge 0$$

Ferns et al. (2004) showed that by applying (1) iteratively, the least fixed point $e_{fix}$ can be obtained, and that $s$ and $u$ are bisimilar if and only if $e_{fix}(s,u) = 0$. In other words, bisimulation is the kernel of this metric.

## 3 Lax bisimulation

In many cases of practical interest, actions with exactly the same label may not match, but the environment may contain symmetries and other types of special structure, which may allow correspondences between *different* actions at certain states. For example, consider the environment in Figure 1. Because of symmetry, going south in state N6 is "equivalent" to going north in state S6. However, no two states are bisimilar. Recent work in process algebra has rethought the definition of bisimulation to allow certain distinct actions to be essentially equivalent (Arun-Kumar, 2006). Here, we define lax bisimulation in the context of MDPs.

**Definition 1.** A relation $B$ is a *lax (probabilistic) bisimulation relation* if whenever $sBu$ we have that: $\forall a \; \exists b$ such that $R(s,a) = R(u,b)$ and for all $B$-closed sets $X$ we have that $Pr(X|s,a) = P(X|u,b)$, and vice versa. The *lax bisimulation* $\sim$ is the union of all the lax bisimulation relations.

It is easy to see that $B$ is an equivalence relation and we denote the equivalence classes of $S$ by $S/B$. Note that the definition above assumes that any action can be matched by any other action. However, the set of actions that can be used to match another action can be restricted based on prior knowledge.

Lax bisimulation is very closely related to the idea of MDP homomorphisms (Ravindran & Barto, 2003). We now formally establish this connection.

**Definition 2.** (Ravindran & Barto, 2003) A *MDP homomorphism h* from $M = \langle S,A,P,R \rangle$ to $M' = \langle S',A',P',R' \rangle$ is a tuple of surjections $\langle f, \{g_s : s \in S\} \rangle$ with $h(s,a) = (f(s),g_s(a))$, where $f : S \to S'$ and $g_s : A \to A'$ such that $R(s,a) = R'(f(s),g_s(a))$ and $P(s,a,f^{-1}(f(s'))) = P'(f(s),g_s(a),f(s'))$

Hence, a homomorphism puts in correspondence states, and has a state-dependent mapping between actions as well. We now show that homomorphisms are identical to lax probabilistic bisimulation.

**Theorem 3.** *Two states s and u are bisimilar if and only if they are related by some MDP homomorphism $\langle f, \{g_s : s \in S\} \rangle$ in the sense that $f(s) = f(u)$.*

**Proof:** For the first direction, let $h$ be a MDP homomorphism and define the relation $B$ such that $sBu$ iff $f(s) = f(u)$. Since $g_u$ is a surjection to $A$, there must be some $b \in A$ with $g_u(b) = g_s(a)$. Hence,

$$R(s,a) = R'(f(s),g_s(a)) = R'(f(u),g_u(b)) = R(u,b)$$

Let $X$ be a non-empty $B$-closed set such that $f^{-1}(f(s')) = X$ for some $s'$. Then:

$$P(s,a,X) = P'(f(s),g_s(a),f(s')) = P'(f(u),g_u(b),f(s')) = P(u,b,X)$$

so $B$ is a lax bisimulation relation.

For the other direction, let $B$ be a lax bisimulation relation. We will construct an MDP homomorphism in which $sBu \implies f(s) = f(u)$. Consider the partition $S/B$ induced by the equivalence relation $B$ on set $S$. For each equivalence class $X \in S/B$, we choose a representative state $s_X \in X$ and define $f(s_X) = s_X$ and $g_{s_X}(a) = a, \forall a \in A$. Then, for any $s \sim s_X$, we define $f(s) = s_X$. From definition 1, we have that $\forall a \exists b$ s.t. $Pr(X'|s,a) = Pr(X'|s_X,b), \forall X' \in S/B$. Hence, we set $g_s(a) = b$. Then, we have:

$$P'(f(s),g_s(a),f(s')) = P'(f(s_X),b',f^{-1}(f(s'))) = P(s_X,b,f^{-1}(f(s'))) = P(s,a,f^{-1}(f(s')))$$

Also, $R'(f(s),g_s(a)) = R'(f(s_X),b) = R(s_X,a)$. Hence, we constructed a homomorphism. $\diamond$

## 4 A metric for lax bisimulation

We will now define a lax bisimulation metric for measuring similarity between state-action pairs, following the approach used by Ferns et al. (2004) for defining the bisimulation metric between states. We want to say that states $s$ and $u$ are close exactly when every action of one state is close to *some* action available in the other state. In order to capture this meaning, we first define similarity between state-action pairs, then we lift this to states using the Hausdorff metric (Munkres, 1999).

**Definition 4.** Let $c_r, c_p \geq 0$ be constants with $c_r + c_p \leq 1$. Given a 1-bounded semi-metric $d$ on $S$, the metric $\delta(d) : S \times A \to [0,1]$ is defined as follows:

$$\delta(d)((s,a),(u,b)) = c_r |R(s,a) - R(u,b)| + c_p K(d)(P(s,a,\cdot), P(u,b,\cdot))$$

We now have to measure the distance between the set of of actions at state $s$ and the set of actions at state $u$. Given a metric between pairs of points, the Hausdorff metric can be used to measure the distance between *sets of points*. It is defined as follows.

**Definition 5.** Given a finite 1-bounded metric space $(\mathcal{M}, d)$, let $\mathcal{P}(\mathcal{M})$ be the set of compact spaces (e.g. closed and bounded in $\mathbb{R}$). The *Hausdorff metric* $H(d) : \mathcal{P}(\mathcal{M}) \times \mathcal{P}(\mathcal{M}) \to [0,1]$ is defined as:

$$H(d)(X,Y) = \max(\sup_{x \in X} \inf_{y \in Y} d(x,y), \sup_{y \in Y} \inf_{x \in X} d(x,y))$$

**Definition 6.** Denote $X_s = \{(s,a) | a \in A\}$. Let $\mathcal{M}$ be the set of all semimetrics on $S$. We define the operator $F : \mathcal{M} \to \mathcal{M}$ as $F(d)(s,u) = H(\delta(d))(X_s, X_u)$

We note that the same definition can be applied both for discrete and for compact continuous action spaces. If the action set is compact then $X_s = \{s\} \times A$ is also compact, so the Hausdorff metric is still well defined. For simplicity, we consider the discrete case, so that max and min are defined.

**Theorem 7.** *$F$ is monotonic and has a least fixed point $d_{fix}$ in which $d_{fix}(s,u) = 0$ iff $s \sim u$.*

The proof is similar in flavor to (Ferns et al., 2004) and we omit it for lack of space.

As both $e_{fix}$ and $d_{fix}$ quantify the difference in behaviour between states, it is not surprising to see that they constrain the difference in optimal value. Indeed, the bound below has previously been shown in (Ferns et al., 2004) for $e_{fix}$, but we also show that our metric $d_{fix}$ is tighter.

**Theorem 8.** *Let $e_{fix}$ be the metric defined in (Ferns et al., 2004). Then we have:*

$$c_r |V^*(s) - V^*(u)| \leq d_{fix}(s,u) \leq e_{fix}(s,u)$$

*Proof:* We show via induction on $n$ that for the sequence of iterates $V_n$ encountered during value iteration, $c_r |V_n(s) - V_n(u)| \leq d_{fix}(s,u) \leq e_{fix}(s,u)$, and then the result follows by merely taking limits.

For the base case note that $c_r |V_0(s) - V_0(u)| = d_0(s,u) = e_0(s,u) = 0$.

Assume this holds for $n$. By the monotonicity of $F$, we have that $F(d_n)(s,u) \leq F(e_n)(s,u)$. Now, for any $a$, $\delta(e_n)((s,a),(u,a)) \leq G(e_n)(s,u)$, which implies:

$$
\begin{aligned}
F(e_n)(s,u) &\leq \max(\max_a \delta(e_n)((s,a),(u,a)), \max_b \delta(e_n)((s,b),(u,b)) \\
&\leq \max(\max_a G(e_n)(s,u), G(e_n)(s,u)) = G(e_n)(s,u)
\end{aligned}
$$

so $d_{n+1} \leq e_{n+1}$ Without loss of generality, assume that $V_{n+1}(s) > V_{n+1}(u)$. Then:

$$
\begin{aligned}
c_r |V_{n+1}(s) - V_{n+1}(u)| &= c_r |\max_a (R(s,a) + \gamma \sum_{s'} P(s,a,s') V_n(s')) - \max_b (R(u,b) + \gamma \sum_{s'} P(u,b,s') V_n(s'))| \\
&= c_r |(R(s,a') + \gamma \sum_{s'} P(s,a',s') V_n(s')) - (R(t,b') + \gamma \sum_{s'} P(u,b',s') V_n(s'))| \\
&= c_r \min_b |(R(s,a') + \gamma \sum_{s'} P(s,a',s') V_n(s')) - (R(u,b) + \gamma \sum_{s'} P(u,b,s') V_n(s'))| \\
&\leq c_r \max_a \min_b |(R(s,a) + \gamma \sum_{s'} P(s,a,s') V_n(s')) - (R(t,b) + \gamma \sum_{s'} P(u,b,s') V_n(s'))| \\
&\leq \max_a \min_b (c_r |R(s,a) - R(u,b)| + c_p |\sum_{s'} (P(s,a,s') - P(u,b,s')) \frac{c_r \gamma}{c_p} V_n(s')|)
\end{aligned}
$$

Now since $\gamma \leq c_p$, we have $0 \leq \frac{c_r \gamma}{c_p} V_i(s') \leq \frac{(1-c_p)\gamma}{c_p(1-\gamma)} \leq 1$ and by the induction hypothesis

$$\frac{c_r \gamma}{c_p} V_n(s) - \frac{c_r \gamma}{c_p} V_n(u) \leq c_r |V_n(s) - V_n(u)| \leq d_n(s,u)$$

So $\{\frac{c_r \gamma}{c_p} V_n(s') : s' \in S\}$ is a feasible solution to the LP for $K(d_n)(P(s,a), P(t,b))$. We then continue the inequality: $c_r |V_{n+1}(s) - V_{n+1}(u)| \leq \max_a \min_b (c_r |R(s,a) - R(u,b)| + c_p K(d_n)(P(s,a), P(u,b))) = F(d_n)(s,u) = d_{n+1}(s,u) \diamond$

# 5 State aggregation

We now show how we can use this notion of lax bisimulation metrics to construct approximate MDP homomorphisms. First, if we have an MDP homomorphism, we can use it to provide a state space aggregation, as follows.

**Definition 9.** Given a MDP $M$ and a homomorphism, an aggregated MDP $M'$ is given by $(S', A, \{P(C, a, D) : a \in A; C, D \in S'\}, \{R(C, a) : a \in A, C \in S'\}, \rho, g_s : s \in S)$ where $S'$ is a partition of $S$, $\rho : S \to S'$ maps states to their aggregates, each $g_s : A \to A$ relabels the action set and we have that $\forall C, D \in S'$ and $a \in A$,

$$P(C, a, D) = \frac{1}{|C|} \sum_{s \in C} P(s, g_s(a), D) \text{ and } R(C, a) = \frac{1}{|C|} \sum_{s \in C} R(s, g_s(a))$$

Note that all the states in a partition have actions that are relabelled specifically so they can exactly match each other's behaviour. Thus, a policy in the aggregate MDP can be lifted to the original MDP by using this relabeling.

**Definition 10.** If $M'$ is an aggregation of MDP $M$ and $\pi'$ is a policy in $M'$, then the lifted policy is defined by $\pi(s) = g_s(\pi'(s'))$.

Using a lax bisimulation metric, it is possible to choose appropriate re-labelings so that states within a partition can approximately match each other's actions.

**Definition 11.** Given a lax bisimulation metric $d$ and a MDP $M$, we say that an aggregated MDP $M'$ is $d$-consistent if each aggregated class $C$ has a state $s \in C$, called the representative of $C$, such that:

$$\forall u \in C, \delta(d)((s, g_s(a)), (u, g_u(a))) \leq F(d)(s, u)$$

When the re-labelings are chosen in this way, we can solve for the optimal value function of the aggregated MDP and be assured that for each state, its true optimal value is close to the optimal value of the partition in which it is contained.

**Theorem 12.** *If $M'$ is a $d_\zeta$-consistent aggregation of a MDP $M$ and $n \leq \zeta$, then $\forall s \in S$ we have:*

$$c_r |V_n(\rho(s)) - V_n(s)| \leq m(\rho(s)) + M \sum_{k=1}^{n-1} \gamma^{n-k}.$$

*where $m(C) = 2 \max_{u \in C} d_\zeta(s', u)$, $s'$ denotes the representative state of $C$ and $M = \max_C m(C)$. Furthermore, if $\pi'$ is a policy in $M'$ and $\pi$ is the corresponding lifted policy in $M$, then:*

$$c_r |V_n^{\pi'}(\rho(s)) - V_n^{\pi}(s)| \leq m(\rho(s)) + M \sum_{k=1}^{n-1} \gamma^{n-k}$$

*Proof:* $|V_{n+1}(\rho(s)) - V_{n+1}(s)| =$

$$= |\max_a (R(\rho(s), a) + \gamma \sum_{D \in S'} P(\rho(s), a, D) V_n(D)) - \max_a (R(s, a) + \gamma \sum_{s'} P(s, a, s') V_n(s'))|$$

$$\leq \frac{1}{|\rho(s)|} \sum_{u \in \rho(s)} \max_a \left( |R(u, g_u(a)) - R(s, g_s(a))| + \gamma | \sum_{D \in S'} P(u, g_u(a), D) V_n(D) - \sum_{s'} P(s, g_s(a), s') V_n(s')| \right)$$

$$\leq \frac{1}{|\rho(s)|} \sum_{u \in \rho(s)} \max_a \left( |R(u, g_u(a)) - R(s, g_s(a))| + \gamma | \sum_{s'} (P(u, g_u(a), s') V_n(\rho(s')) - P(s, g_s(a), s') V_n(s'))| \right)$$

$$\leq \frac{1}{|\rho(s)|} \sum_{u \in \rho(s)} \max_a (|R(u, g_u(a)) - R(s, g_s(a))| + \gamma | \sum_{s'} (P(u, g_u(a), s') - P(s, g_s(a), s')) V_n(s')$$

$$+ \gamma | \sum_{s'} P(u, g_u(a), s') (V_n(\rho(s')) - V_n(s'))|) \leq \frac{1}{c_r |\rho(s)|} \sum_{u \in \rho(s)} \max_a (c_r |R(s, g_s(a)) - R(u, g_u(a))|$$

$$+ c_p | \sum_{s'} (P(u, g_u(a), s') - P(s, g_s(a), s')) \frac{c_r \gamma}{c_p} V_n(s')|) + \frac{\gamma}{|\rho(s)|} \sum_{u \in \rho(s)} \max_a \sum_{s'} P(u, g_u(a), s') |V_n(\rho(s')) - V_n(s')|$$

From Theorem 8, we know that $\{\frac{c_r\gamma}{c_p}V_n(s') : s' \in S\}$ is a feasible solution to the primal LP for $K(d_n)(P(s,g_s(a)),P(u,g_u(a)))$. Let $z$ be the representative used for $\rho(s)$. Then we can continue as follows:

$$\leq c_r|R(s,g_s(a)) - R(u,g_u(a))| + c_pK(d_n)(P(s,g_s(a)),P(u,g_u(a)))$$
$$\leq c_r|R(s,g_s(a)) - R(u,g_u(a))| + c_pK(d_\zeta)(P(s,g_s(a)),P(u,g_u(a)))$$
$$\leq c_r|R(s,g_s(a)) - R(z,g_z(a))| + c_pK(d_\zeta)(P(s,g_s(a)),P(z,g_z(a)))$$
$$+ c_r|R(z,g_z(a)) - R(u,g_u(a))| + c_pK(d_\zeta)(P(z,g_z(a)),P(u,g_u(a))) = d_\zeta(s,z) + d_\zeta(z,u) \leq m(\rho(s))$$

We continue with the original inequality using these two results:

$$\leq \quad \frac{1}{c_r}\sum_{u\in\rho(s)}(c_r|R(s,g_s(a)) - R(u,g_u(a))| + c_pK(d_n)(P(s,g_s(a)),P(u,g_u(a))))$$

$$+ \quad \frac{\gamma}{|\rho(s)|}\sum_{u\in\rho(s)}\max_a\sum_{s'}P(u,g_u(a),s')\max_{s''}|V_n(\rho(s'')) - V_n(s'')|$$

$$\leq \quad \frac{1}{c_r|\rho(s)|}\sum_{u\in\rho(s)}m(\rho(s)) + \gamma\max_{s'}|V_n(\rho(s')) - V_n(s')| \leq \frac{m(\rho(s))}{c_r} + \gamma\max_{s''}\left(\frac{m(\rho(s))}{c_r} + M\sum_{k=1}^{n-1}\gamma^{n-k}\right)$$

$$\leq \quad \frac{1}{c_r}\left(m(\rho(s)) + \gamma\max_{s'}m(\rho(s')) + M\sum_{k=1}^{n-1}\gamma^{n+1-k}\right) \leq \frac{1}{c_r}\left(m(\rho(s)) + M\sum_{k=1}^{n}\gamma^{(n+1)-k}\right)$$

The second proof is nearly identical except that instead of maximizing over actions, the action selected by the policy, $a = \pi'(\rho(s))$, and the lifted policy, $g_s(a) = \pi(s)$ are used. ⋄

By taking limits we get the following theorem:

**Theorem 13.** *If $M'$ is a $d_{fix}$-consistent aggregation of a MDP M, then $\forall s \in S$ we have:*

$$c_r|V^*(\rho(s)) - V^*(s)| \leq m(\rho(s)) + \frac{\gamma}{1-\gamma}M$$

*Furthermore, if $\pi'$ is any policy in $M'$ and $\pi$ is the lifted policy to M then*

$$c_r|V^{\pi'}(\rho(s)) - V^\pi(s)| \leq m(\rho(s)) + \frac{\gamma}{1-\gamma}M$$

*where $m(C) = 2\max_{u\in C}d_{fix}(s',u)$, $s'$ is the representative state of C and $M = \max_C m(C)$.*

One appropriate way to aggregrate states is to choose some desired error bound $\varepsilon > 0$ and ensure that the states in each partition are within an $\varepsilon$-ball. A simple way to do this is to pick states and random and add to a partition each state within the $\varepsilon$-ball. Of course, better clustering heuristics can be used here as well.

It has been noted that when the above condition holds, then under the unlaxed bisimulation metric $e_{fix}$, we can be assured that for each state $s$, $|V^*(\rho(s)) - V(s)|$ is bounded by $\frac{2\varepsilon}{c_r(1-\gamma)}$. The theorem above shows that under the lax bisimulation metric $d_{fix}$ this difference is actually bounded by $\frac{4\varepsilon}{c_r(1-\gamma)}$. However, as we illustrate in the next section. a massive reduction in the size of the state space can be achieved by moving from $e_{fix}$ to $d_{fix}$, even when using $\varepsilon' = \frac{\varepsilon}{2}$.

For large systems, it might not be feasible to compute the metric $e_{fix}$ in the original MDP. In this case, we might want to use some sort of heuristic or prior knowledge to create an aggregation. Ravindran & Barto (2003) provided, based on a result from Whitt (1978), a bound on the difference in values between the optimal policy in the aggregated MDP and the lifted policy in the original MDP. We now show that our metric can be used to tighten this bound.

**Theorem 14.** *If $M'$ is an aggregation of a MDP M, $\pi'$ is an optimal policy in $M'$, $\pi$ is the policy lifted from $\pi'$ to M and $d'_{fix}$ corresponds to our metric computed on $M'$, then*

$$|V^\pi(s) - V^{\pi'}(\rho(s))| \leq \frac{2}{1-\gamma}\max_{s,a}|R(s,g_s(a)) - R(\rho(s),a)| + \frac{\gamma}{c_r}\max_{s,a}K(d'_{fix})(P(s,g_s(a)),P(\rho(s),a))$$

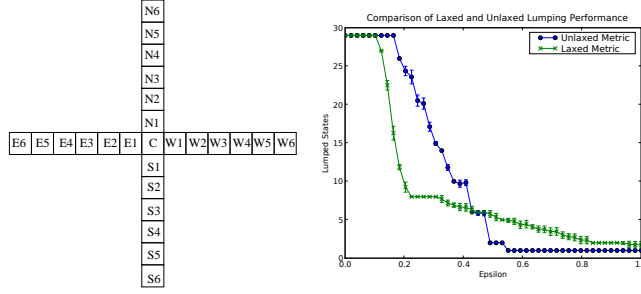

Figure 1: Example environment exhibiting symmetries (left). Aggregation performance (right)

*Proof:* We have:

$$|V^{\pi}(s) - V^{\pi'}(\rho(s))| \leq \frac{2}{1-\gamma} \max_{s,a} |R(s,g_s(a)) - R(\rho(s),a) + \gamma \sum_C (P(s,g_s(a),C) - P(\rho(s),a,C))V^{\pi'}(C)|$$

$$\leq \frac{2}{1-\gamma} \max_{s,a} |R(s,g_s(a)) - R(\rho(s),a)| + \gamma \max_{s,a} |\sum_C (P(s,g_s(a),C) - P(\rho(s),a,C))V^{\pi'}(C)|$$

$$\leq \frac{2}{1-\gamma} \max_{s,a} |R(s,g_s(a)) - R(\rho(s),a)| + \max_{s,a} \frac{\gamma}{c_r} K(d'_{fix})(P(s,g_s(a)), P(\rho(s),a))$$

The first inequality originally comes from (Whitt, 1978) and is applied to MDPs in (Ravindran & Barto, 2003). The last inequality holds since $\pi'$ is an optimal policy and thus by Theorem 8 we know that $\{\frac{V^{\pi'}(C)}{c_r} : C \in S'\}$ is a feasible solution. ◇

As a corrolary, we can get the same bound as in (Ravindran & Barto, 2003) by bounding the Kantorovich by the total variation metric.

**Definition 15.** Given two finite distributions $P$ and $Q$, the total variation metric $TV(P,Q)$ is defined as: $TV(P,Q) = \sum_s \frac{1}{2}|P(s) - Q(s)|$

**Corollary 16.** *Let $\Delta = \max_{C,a} R(C,a) - \min_{C,a} R(C,a)$ be the maximum difference in rewards in the aggregated MDP. Then:*

$$|V^{\pi}(s) - V^{\pi}(\rho(s))| \leq \frac{2}{1-\gamma} \left( \max_{s,a} |R(s,g_s(a)) - R(\rho(s),a)| + \frac{\gamma}{1-\gamma} \Delta \cdot TV(P(s,g_s(a)), P(\rho(s),a)) \right)$$

*Proof:* This follows from the fact that:

$$\max_{C,D} d'_{fix}(C,D) \leq c_r \Delta + c_p \max_{C,D} d'_{fix}(C,D) \cdots \leq \frac{c_r \Delta}{1-c_p} \leq \frac{c_r \Delta}{1-\gamma}$$

and using the total variation as an approximation (Gibbs & Su, 2002), we have:

$$K(d'_{fix})(P(s,g_s(a)), P(\rho(s),a)) \leq \max_{C,D} d'_{fix}(C,D) \cdot TV(P(s,g_s(a)), P(\rho(s),a)) \quad \diamond$$

## 6 Illustration

Consider the cross-shaped MDP displayed in Figure 1. There is a reward of 1 in the center and the probability of the agent moving in the intended direction is 0.8. For a given $\varepsilon$, we used the random partitioning algorithm outlined earlier to create a state aggregation. The graph plots the size of the aggregated MDPs obtained against $\varepsilon$, using the lax and the non-lax bisimulation metrics. In the case of the lax metric, we used $\varepsilon' = \varepsilon/2$ to compensate for the factor of 2 difference in the error bound. It is very revealing that the number of partitions drops very quickly and levels at around 6 or 7 for our algorithm. This is because the MDP is collapsing to a state space close to the natural choice of $\{\{C\}\} \cup \{\{Ni,Si,Wi,Ei\} : i \in \{1,2,3,4,5,6\}\}$. Under the unlaxed metric, this is not likely to occur, and thus the first states to be partitioned together are the ones neighbouring each other (which can actually have quite different behaviours).

# 7 Discussion and future work

We defined a metric for measuring the similarity of state-action pairs in a Markov Decision Process and used it in an algorithm for constructing approximate MDP homomorphisms. Our approach works significantly better than the bisimulation metrics of Ferns et al., as it allows capturing different regularities in the environment. The theoretical bound on the error in the value function presented in (Ravindran & Barto, 2004) can be derived using our metric.

Although the metric is potentially expensive to compute, there are domains in which having an accurate aggregation is worth it. For example, in mobile device applications, one may have big computational resources initially to build an aggregation, but may then insist on a very coarse, good aggregation, to fit on a small device. The metric can also be used to find subtasks in a larger problem that can be solved using controllers from a pre-supplied library. For example, if a controller is available to navigate single rooms, the metric might be used to lump states in a building schematic into "rooms". The aggregate MDP can then be used to solve the high level navigational task using the controller to navigate specific rooms.

An important avenue for future work is reducing the computational complexity of this approach. Two sources of complexity include the quadratic dependence on the number of actions, and the evaluation of the Kantorovich metric. The first issue can be addressed by sampling pairs of actions, rather than considering all possibilities. We are also investigating the possibility of replacing the Kantorovich metric (which is very convenient from the theoretical point of view) with a more practical approximation. Finally, the extension to continuous states is very important. We currently have preliminary results on this issue, using an approach similar to (Ferns et al, 2005), which assumes lower-semi-continuity of the reward function. However, the details are not yet fully worked out.

**Acknowledgements:** This work was funded by NSERC and CFI.

# References

Arun-Kumar, S. (2006). On bisimilarities induced by relations on actions. *SEFM '06: Proceedings of the Fourth IEEE International Conference on Software Engineering and Formal Methods* (pp. 41–49). Washington, DC, USA: IEEE Computer Society.

Ferns, N., Castro, P. S., Precup, D., & Panangaden, P. (2006). Methods for computing state similarity in Markov Decision Processes. *Proceedings of the 22nd UAI*.

Ferns, N., Panangaden, P., & Precup, D. (2004). Metrics for finite markov decision processes. *Proceedings of the 20th UAI* (pp. 162–169).

Ferns, N., Panangaden, P., & Precup, D. (2005). Metrics for markov decision processes with infinite state spaces. *Proceedings of the 21th UAI* (pp. 201–209).

Gibbs, A., & Su, F. (2002). On choosing and bounding probability metrics.

Givan, R., Dean, T., & Greig, M. (2003). Equivalence notions and model minimization in Markov Decision Processes. *Artificial Intelligence*, *147*, 163–223.

Larsen, K. G., & Skou, A. (1991). Bisimulation through probabilistic testing. *Inf. Comput.*, *94*, 1–28.

Li, L., Walsh, T. J., & Littman, M. L. (2006). Towards a unified theory of state abstraction for MDPs. *Proceedings of the International Symposium on Artificial Intelligence and Mathematics*.

Milner, R. (1995). *Communication and concurrency*. Prentice Hall International (UK) Ltd.

Munkres, J. (1999). *Topology*. Prentice Hall.

Puterman, M. L. (1994). *Markov decision processes: discrete stochastic dynamic programming*. Wiley.

Ravindran, B., & Barto, A. G. (2003). Relativized options: Choosing the right transformation. *Proceedings of 20th ICML* (pp. 608–615).

Ravindran, B., & Barto, A. G. (2004). Approximate homomorphisms: A framework for non-exact minimization inn Markov Decision Processes. *Proceedings of the Fifth International Conference on Knowledge Based Computer Systems*.

Whitt, W. (1978). Approximations of dynamic programs i. *Mathematics of Operations Research*, *3*, 231–243.

Wolfe, A. P., & Barto, A. G. (2006). Decision tree methods for finding reusable MDP homomorphisms. *Proceedings of AAAI*.
